# Constructing Topological Maps using Markov Random Fields and Loop-Closure Detection

**Roy Anati    Kostas Daniilidis**
GRASP Laboratory
Department of Computer and Information Science
University of Pennsylvania
Philadelphia, PA 19104
`{royanati,kostas}@cis.upenn.edu`

## Abstract

We present a system which constructs a topological map of an environment given a sequence of images. This system includes a novel image similarity score which uses dynamic programming to match images using both the appearance and relative positions of local features simultaneously. Additionally, an MRF is constructed to model the probability of loop-closures. A locally optimal labeling is found using Loopy-BP. Finally we outline a method to generate a topological map from loop closure data. Results, presented on four urban sequences and one indoor sequence, outperform the state of the art.

## 1 Introduction

The task of generating a topological map from video data has gained prominence in recent years. Topological representations of routes spanning multiple kilometers are robuster than metric and cognitively more plausible for use by humans. They are used to perform path planning, providing waypoints, and defining reachability of places. Topological maps can correct for the drift in visual odometry systems and can be part of hybrid representations where the environment is represented metrically locally but topologically globally.

We identify two challenges in constructing a topological map from video: how can we say whether two images have been taken from the same place; and how can we reduce the original set of thousands of video frames to a reduced representative set of keyframes for path planning. We take into advantage the fact that our input is video as opposed to an unorganized set of pictures. Video guarantees that keyframes will be reachable to each other but it also provides temporal ordering constraints on deciding about loop closures. The paper has three innovations: We define a novel image similarity score which uses dynamic programming to match images using both the appearance and the layout of the features in the environment. Second, graphical models are used to detect loop-closures which are locally consistent with neighboring images. Finally, we show how the temporal assumption can be used to generate compact topological maps using minimum dominating sets.

We formally define a topological map $T$ as a graph $T = (K, E_T)$, where $K$ is a set of keyframes and $E_T$ edges describing connectivity between keyframes. We will see later that keyframes are representatives of locations. We desire the following properties of $T$:

**Loop closure** For any two locations $i, j \in K$, $E_T$ contains the edge $(i, j)$ if and only if it is possible to reach location $j$ from location $i$ without passing through any other location $k \in K$.

**Compactness** Two images taken at the "same location" *should* be represented by the same keyframe.

**Spatial distinctiveness** Two images from "different locations" *cannot* be represented by the same keyframe.

Note that spatial distinctiveness requires that we distinguish between separate locations, however compactness encourages agglomeration of geographically similar images. This distinction is important, as lack of compactness does not lead to errors in either path planning or visual odometry while breaking spatial distinctiveness does. Our approach to building topological maps is divided into three modules: calculating image similarity, detecting loop closures, and map construction. As defined it is possible to implement each module independently, providing great flexibility in the algorithm selection. We now define the interfaces between each pair of modules.

Starting with $\mathcal{I}$, a sequence of $n$ images, the result of calculating image similarity scores is a matrix $M_{n \times n}$ where $M_{ij}$ represents a relative similarity between images $i$ and $j$. In section 2 we describe how we use local image features to compute the matrix $M$. To detect loop-closures we have to discretize $M$ into a binary decision matrix $D_{n \times n}$ where $D_{ij} = 1$ indicates that images $i$ and $j$ are geographically equivalent and form a loop closure. Section 3 describes the construction of $D$ by defining a Markov Random Field (MRF) on $M$ and perform approximate inference using Loopy Belief Propagation (Loopy-BP). In the final step, the topological map $T$ is generated from $D$. We calculate the set of keyframes $K$ and their associated connectivity $E_T$ using the minimum dominating set of the graph represented by $D$ (Section 4).

**Related Work** The state of the art in topological mapping of images is the FAB-MAP [8] algorithm. FAB-MAP uses bag of words to model locations using a generative appearance approach that models dependencies and correlations between visual words rendering FAB-MAP extremely successful in dealing with the challenge of perceptual aliasing (different locations sharing common visual characteristics). Its implementation outperforms any other in speed averaging an intra-image comparison of less than 1ms. Bayesian inference is also used in [1] where bags of words on local image descriptors model locations whose consistency is validated with epipolar geometry. Ranganathan et al. [14] incorporate both odometry and appearance and maintain several hypotheses of topological maps. Older approaches like ATLAS [5] and Tomatis et al. [17] define maps on two levels, creating global (topological) maps by matching independent local (metric) data and combining loop -closure detection with visual SLAM (Self Localization and Mapping). The ATLAS framework [5] matches local maps through the geometric structures defined by their 2D schematics whose correspondences define loop-closures. Tomatis et al [17] detect loop closures by examining the modality of the robot position's density function (PDF). A PDF with two modes traveling in sync is the result of a missed loop-closure, which is identified and merged through backtracking.

Approaches like [3] [19] [18] and [9] represent the environment using only an image similarity matrix. Booij et al [3] use the similarity matrix to define a weighted graph for robot navigation. Navigation is conducted on a node by node basis, using new observations and epipolar geometry to estimate the direction of the next node. Valgren et al [19] avoid exhaustively computing the similarity matrix by searching for and sampling cells which are more likely to describe existing loop-closures. In [18], they employ exhaustive search, but use spectral clustering to reduce the search space incrementally when new images are processed. Fraundoerfer et al [9] use hierarchical vocabulary trees [13] to quickly compute image similarity scores. They show improved results by using feature distances to weigh the similarity score. In [15] a novel image feature is constructed from patches centered around vertical lines from the scene (radial lines in the image). These are then used to track the bearing of landmarks and localize the robot in the environment. Goedeme [10] proposes 'invariant column segments' combined with color information to compare images. This is followed by agglomerative clustering of images into locations. Potential loop-closures are identified within clusters and confirmed u sing Dempster-Shafer probabilities.

Our approach advances the state of the art by using a powerful image alignment score without employing full epipolar geometry, and more robust loop colsure detection by applying MRF inference on the similarity matrix. It is together with [4] the only video-based approach that provides a greatly reduced set of nodes for the final topological representation, making thus path planning tractable.

## 2  Image similarity score

For any two images $i$ and $j$, we calculate the similarity score $M_{ij}$ in three steps: generate image features, sort image features into sequences, calculate optimal alignment between both sequences. To detect and generate image features we use Scale Invariant Feature Transform (SIFT) [12]. SIFT was selected as it is invariant to rotation and scale, and partially immune to other affine transformations.

**Feature sequences**  Simply matching the SIFT features by value [12] yields satisfactory results (see later in figure 2). However, to mitigate perceptual aliasing, we take advantage of the fact that features represent real world structures with fixed spatial arrangements and therefore the similarity score should take their relative positions into account. A popular approach, employed in [16], is to enforce scene rigidity by validating the epipolar geometry between two images. This process, although extremely accurate, is expensive and very time-consuming. Instead, we make the assumption that the gravity vector is known so that we can split image position into bearing and elevation and we take into account only the bearing of each feature. Sorting the features by their bearing, results in ordered sequences of SIFT features. We then search for an optimal alignment between pairs of sequences, incorporating both the value and ordering of SIFT features into our similarity score.

**Sequence alignment**  To solve for the optimal alignment between two ordered sequences of features we employ dynamic programming. Here a match between two features, $f_a$ and $f_b$, occurs if their $L_1$ norm is below a threshold, $Score(a, b) = 1$ if $|f_a - f_b|_1 < t_{match}$. A key aspect to dynamic programming is the enforcement of the ordering constraint. This ensures that the relative order of features matched is consistent in both sequences, exactly the property desired to ensure consistency between two scene appearances. Since bearing is not given with respect to an absolute orientation, ordering is meant only cyclically, which can be handled easily in dynamic programming by replicating one of the input sequences. Modifying the first and last rows of the score matrix to allow for arbitrary start and end locations yields the optimal cyclical alignment in most cases. This comes at the cost of allowing one-to-many matches which can result in incorrect alignment scores. The score of the optimal alignment between both sequences of features provides the basis for the similarity score between two images and the entries of the matrix $M$. We calculate the values of $M_{ij}$ for all $i < j - w$. Here $w$ represents a window used to ignore images immediately before/after our query.

## 3  Loop closure-detection using MRF

Using the image similarity measure matrix $M$, we use Markov Random Fields to detect loop-closures. A lattice $H$ is defined as an $n \times n$ lattice of binary nodes where a node $v_{i,j}$ represents the probability of images $i$ and $j$ forming a loop-closure. The matrix $M$ provides an initial estimate of this value. We define the factor $\phi_{i,j}$ over the node $v_{i,j}$ as follows: $\phi_{i,j}(1) = M_{ij}/F$ and $\phi_{i,j}(0) = 1 - \phi_{i,j}(1)$ where $F = \max(M)$ is used to normalize the values in $M$ to the range $[0, 1]$. Loops closures in the score matrix $M$ appear as one of three possible shapes. In an intersection the score matrix contains an ellipse. A parallel traversal, when a vehicle repeats part of its trajectory, is seen as a diagonal band. An inverse traversal, when a vehicle repeats a part of its trajectory in the opposite direction, is an inverted diagonal band. The length and thickness of these shapes vary with the speed of the vehicle (see figure 1 for examples of these shapes). Therefore we define lattice $H$ with eight way connectivity, as it better captures the structure of possible loop closures. As adjacent nodes in $H$ represent sequential images in the sequence, we expect significant overlap in their content. So two neighboring nodes (in any orientation), are expected to have similar scores. Sudden changes occur when either a loop is just closed (sudden increase) or when a loop closure is complete (sudden decrease) or due to noise caused by a sudden occlusion in one of the scenes. By imposing smoothness on the labeling we capture loop closures while discarding noise. Edge potentials are therefore defined as Gaussians of differences in $M$. Letting $G(x, y) = e^{-\frac{(x-y)^2}{\sigma^2}}$, $k = \{i - 1, i, i + 1\}$ and $l = \{j - 1, j, j + 1\}$ then

$$\phi_{i,j,k,l}(0,0) = \phi_{i,j,k,l}(1,1) = \quad \alpha \cdot G\left(M_{ij}, M_{kl}\right)$$
$$\phi_{i,j,k,l}(0,1) = \phi_{i,j,k,l}(1,0) = \quad\quad 1,$$

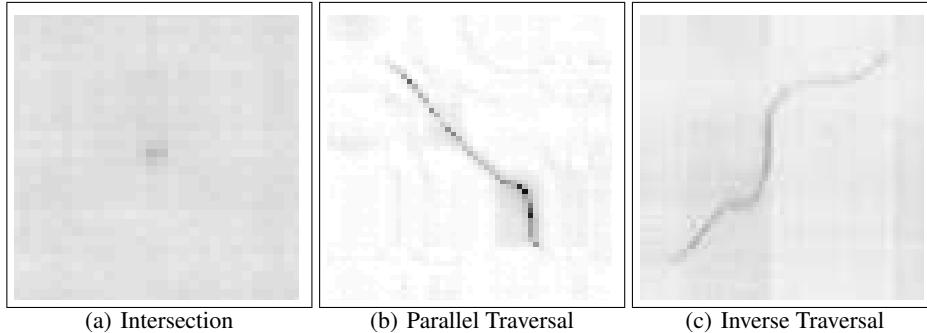

|(a) Intersection|(b) Parallel Traversal|(c) Inverse Traversal|

Figure 1: A small ellipse resulting from an intersection (a) and two diagonal bands from a parallel (b) and inverse (c) traversals. All extracted from a score matrix $M$.

where $1 \leq \alpha$ (we ignore the case when both $k = i$ and $j = l$). Overall, $H$ models a probability distribution over a labeling $v \in \{1, 0\}^{n \times n}$ where:

$$P(v) = \frac{1}{Z} \prod_{i,j \in [1,n]} \phi_{i,j}(v_{i,j}) \prod_{i,j \in [1,n]} \prod_{k=[i-1,i+1]} \prod_{l=[j-1,j+1]} \phi_{i,j,k,l}(v_{i,j}, v_{k,l})$$

In order to solve for the MAP labeling of $H$, $v^* = \arg\max_v P(v)$, the lattice must first be transformed into a cluster graph $C$. This transformation allows us to model the beliefs of all factors in the graph and the messages being passed during inference. We model every node and every edge in $H$ as a node in the cluster graph $C$. An edge exists between two nodes in the cluster graph if the relevant factors share variables. In addition this construction presents a two step update schedule, alternating between 'node' clusters and 'edge' clusters as each class only connects to instances of the other. Once defined, a straightforward implementation of the generalized max-product belief propagation algorithm (described in both [2] and [11]) serves to approximate the final labeling. We initialize the cluster graph directly from the lattice $H$ with $\psi_{i,j} = \phi_{i,j}$ for nodes and $\psi_{i,j,k,l} = \phi_{i,j,k,l}$ for edges. The MAP labeling found here defines our matrix $D$ determining whether two images $i$ and $j$ close a loop. Note, that the above MAP labeling is guaranteed to be locally optimal, but is not necessarily consistent across the entire lattice. Generally, finding the globally consistent optimal assignment is NP-hard [11]. Instead, we rely on our definition of $D$, which specifies which pairs of images are equivalent, and our construction in section 4 to generate consistent results.

## 4  Constructing the topological map

Finally the decision matrix $D$ is used to define keyframes $K$ and determine the map connectivity $E_T$. $D$ can be viewed as an adjacency matrix of an undirected graph. Since there is no guarantee that $D$ found through belief propagation is symmetric, we initially treat $D$ as an adjacency matrix for a directed graph, and then remove the direction from all the edges resulting in a symmetric graph $D' = D \vee D^{\mathrm{T}}$. It is possible to use the graph defined by $D'$ as a topological map. However this representation is practically useless because multiple nodes represent the same location. To achieve compactness, $D'$ needs to be pruned while remaining faithful to the overall structure of the environment. Booij [4] achieve this by approximating for the minimum *connected* dominating set. By using the temporal assumption we can remove the connectedness requirement and use minimum dominating set to prune $D'$. We find the keyframes $K$ by finding the minimum dominating set of $D'$. Finding the optimal solution is NP-Complete, however algorithm 1 provides a greedy approximation. This approximation has a guaranteed bound of $H(d_{max})$ (harmonic function of the maximal degree in the graph $d_{max}$) [6].

The dominating set itself serves as our keyframes $K$. Each dominating node $k \in K$ is also associated with the set of nodes it dominates $N_k$. Each set $N_k$ represent images which have the "same location". The sets $\{N_k : k \in K\}$ in conjunction with our underlying temporal assumption are used to connect the map $T$. An edge $(k, j)$ is added if $N_k$ and $N_l$ contain two consecutive images from our sequence, i.e. $(k, j) \in E_T$ if $\exists i$ such that $i \in N_k$ and $i + 1 \in N_l$. This yields our final topological map $T$.

---
**Algorithm 1**: Approximate Minimum Dominating Set
---
  **Input**: Adjacency matric $D'$
  **Output**: $K,\{N_k : k \in K\}$
  $K \leftarrow \emptyset$
  **while** $D'$ *is not empty* **do**
      $k \leftarrow$ node with largest degree
      $K \leftarrow K \cup \{k\}$
      $N_k \leftarrow \{k\} \cup Nb(k)$
      Remove all nodes $N_k$ from matrix $D'$
  **end**
---

## 5  Experiments

The system was applied to five image sequences. Results are shown for the system as described, as well as for FAB-MAP ([8]) and for different methods of calculating image similarity scores.

**Image sets**  Three image sequences, *indoors*, *Philadelphia* and *Pittsburgh*[1] were captured with a Point Gray Research Ladybug camera. The Ladybug is composed of five wide-angle lens camera arranged in circle around the base and one camera on top facing upwards. The resulting output is a sequence of frames each containing a set of images captured by the six cameras. For the outdoor sequences the camera was mounted on top of a vehicle which was driven around an urban setting, in this case the cities of Philadelphia and Pittsburgh. In the indoor sequence, the camera was mounted on a tripod set on a cart and moved inside the building covering the ground and 1st floors. Ladybug images were processed independently for each camera using the SIFT detector and extractor provided in the VLFeat toolbox [20]. The resulting features for every camera were merged into a single set and sorted by their spherical coordinates. The two remaining sequences, *City Centre* and *New College* were captured in an outdoor setting by Cummins [7] from a limited field of view camera mounted on a mobile robot. Table 1 summarizes some basic properties of the sequences we use. All the outdoor sequences were provided with GPS location of the vehicle / robot. For Philadelphia

| Data Set | Length | No. of frames | Camera Type | Format |
|---|---|---|---|---|
| Indoors | Not available | 852 | spherical | raw Ladybug stream file |
| Philadelphia[16] | 2.5km | 1,266 | spherical | raw Ladybug stream file |
| Pittsburgh | 12.5km | 1,256 | spherical | rectified images |
| New College[7] | 1.9km | 1,237 | limited field of view | standard images |
| City Centre[7] | 2km | 1,073 | limited field of view | standard images |

Table 1: Summary of image sequences processed.

and Pittsburgh, these were used to generate ground truth decision matrices using a threshold of 10 meters. Ground truth matrices were provided for New College and City Centre. For the indoor sequence the position of the camera was manually determined using building schematics at an arbitrary scale. A ground truth decision matrix was generated using a manually determined threshold. The entire system was implemented in Matlab with the exception of the SIFT detector and extractor implemented by [20].

**Parameters**  Both the image similarity scores and the MRF contain a number of parameters that need to be set. When calculating the image similarity score, there are five parameters. The first $t_{match}$ is the threshold on th $L_1$ norm at which two SIFT features are considered matched. In addition, dynamic programming requires three parameters to define the score of an optimal alignment: $s_{match}, s_{gap}, s_{miss}$. $s_{match}$ is the value by which the score of an alignment is improved by including correctly matched pairs of features. $s_{gap}$ is the cost of ignoring a feature in the optimal alignment (insertion and deletion), and $s_{miss}$ is the cost of including incorrectly matched pairs (substitution). We use $t_{match} = 1000$, $s_{match} = 1$, $s_{gap} = -0.1$ and $s_{miss} = 0$. Finally we use $w = 30$ as our window size, to avoid calculating similarity scores for images taken within very short time of each

|            | Indoors | Philadelphia | Pittsburgh | City Centre | New College |
|------------|---------|--------------|------------|-------------|-------------|
| Precision  | 91.67%  | 91.72%       | 63.85%     | 97.42%      | 91.57%      |
| Recall     | 79.31%  | 51.46%       | 54.60%     | 40.04%      | 84.35%      |

Table 2: Precision and recall after performing inference.

other. Constructing the MRF requires three parameters, $F$, $\sigma$ and $\alpha$. The normalization factor, $F$, has already been defined as $\max(M)$. The $\sigma$ used in defining edge potentials is $\sigma = 0.05F$ where $F$ is again used to rescale the data in the interval $[0, 1]$. Finally we set $\alpha = 2$ to rescale the Gaussian to favor edges between similarly valued nodes. Inference using loopy belief propagation features two parameters, a dampening factor $\lambda = 0.5$ used to mitigate the effect of cyclical inferencing and $n = 20$, the number of iterations over which to perform inference.

**Results**   In addition to the image similarity score defined above, we also processed the image sequences using alternative similarity measures. We show results for $M_{ij}^{SIFT}$ = number of SIFT matches, $M_{ij}^{REC}$ = number of *reciprocal* SIFT matches (the intersection of matches from image $i$ to image $j$ and from $j$ to $i$). We also show results using FAB-MAP [8]. To process spherical images using FAB-MAP we limited ourselves to using images captured by camera 0 (Directly forwards / backwards). Figure 2 shows precision-recall curves for all sequences and similarity measures. The curves were generated by thresholding the similarity scores. Our method outperforms state of the art in terms of precision and recall in all sequences. The gain from using our system is most pronounced in the *Philadelphia* sequence, where FAB-MAP yields extremely low recall rates. Table 2 shows the results of performing inference on the image similarity matrices. Finally figure 3 shows the topological map resulting from running dominating sets on the decision matrices $D$. We use the ground truth GPS positions for display purposes only. The blue dots represent the locations of the keyframes $K$ with the edges $E_T$ drawn in blue. Red dots mark keyframes which are also loop-closures. For reference, figure 4 provides ground truth maps and loop-closures.

## 6   Outlook

We presented a system that constructs purely topological maps from video sequences captured from moving vehicles. Our main assumption is that the images are presented in a temporally consistent manner. A highly accurate image similarity score is found by a cyclical alignment of sorted feature sequences. This score is then refined via loopy-belief propagation to detect loop-closures. Finally we constructed a topological map for the sequence in question. This map can be used for either path planning or for bundle adjustment in visual SLAM systems. The bottleneck of the system is computing the image similarity score. In some instances, taking over 166 hours to process a single sequence while FAB-MAP [8] accomplishes the same task in 20 minutes. In addition to implementing score calculation with a parallel algorithm (either on a multicore machine or using graphics hardware), we plan to construct approximations to our image similarity score. These include using visual bags of words in a hierarchical fashion [13] and building the score matrix $M$ incrementally [19, 18].

**Acknowledgments**

Financial support by the grants NSF-IIS-0713260, NSF-IIP-0742304, NSF-IIP-0835714, and ARL/CTA DAAD19-01-2-0012 is gratefully acknowledged.

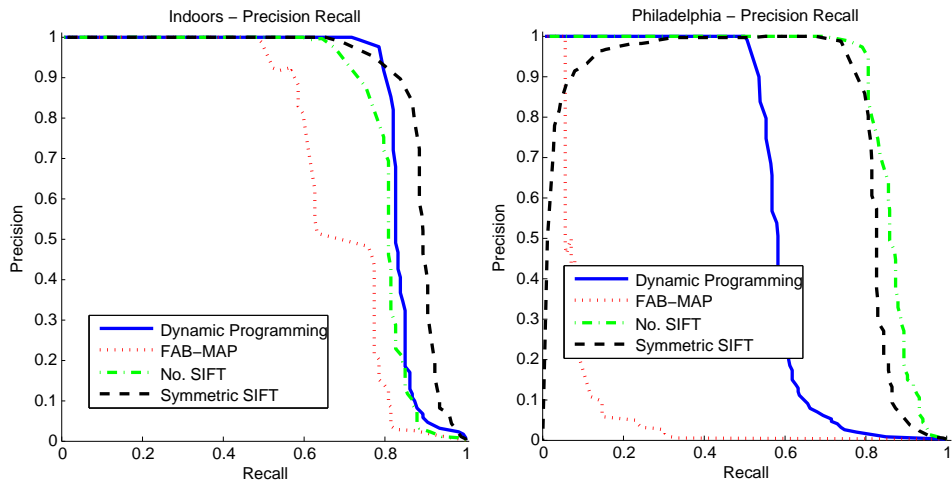

(a) Indoors                      (b) Philadelphia

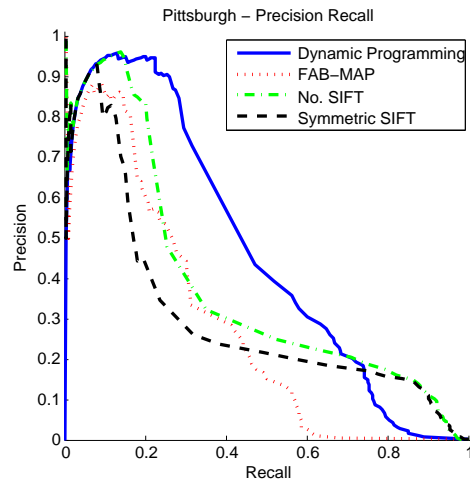

(c) Pittsburgh

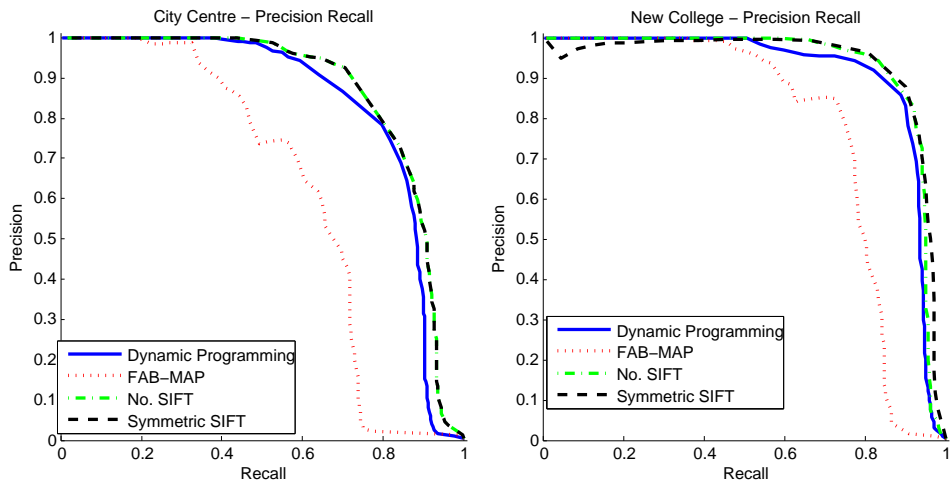

(d) City Centre                    (e) New College

Figure 2: Precision-recall curves for different thresholds on image similarity scores.

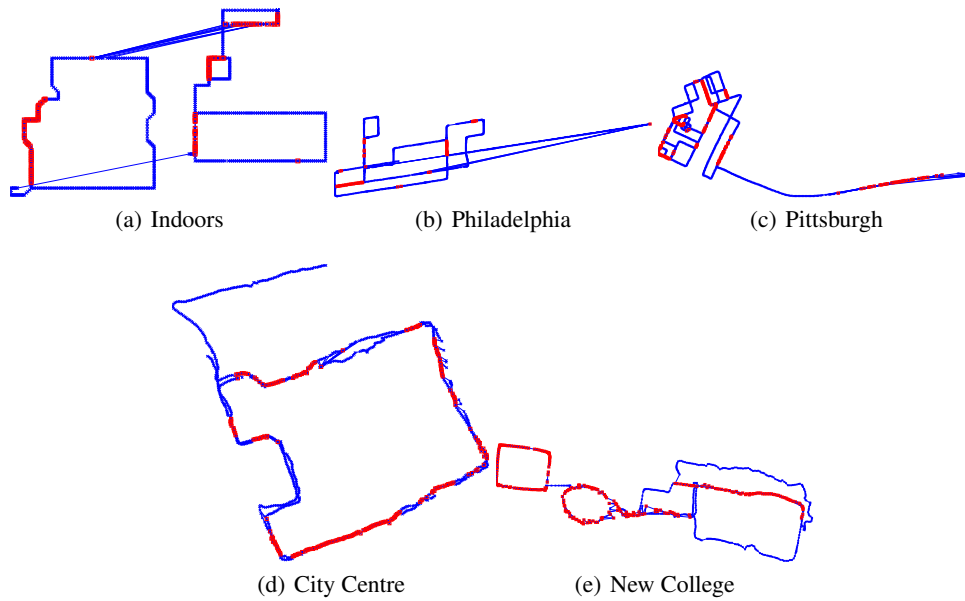

Figure 3: Loop-closures generated using minimum dominating set approximation. Blue dots represent positions of keyframes $K$ with edges $E_T$ drawn in blue. Red dots mark keyframes with loop-closures.

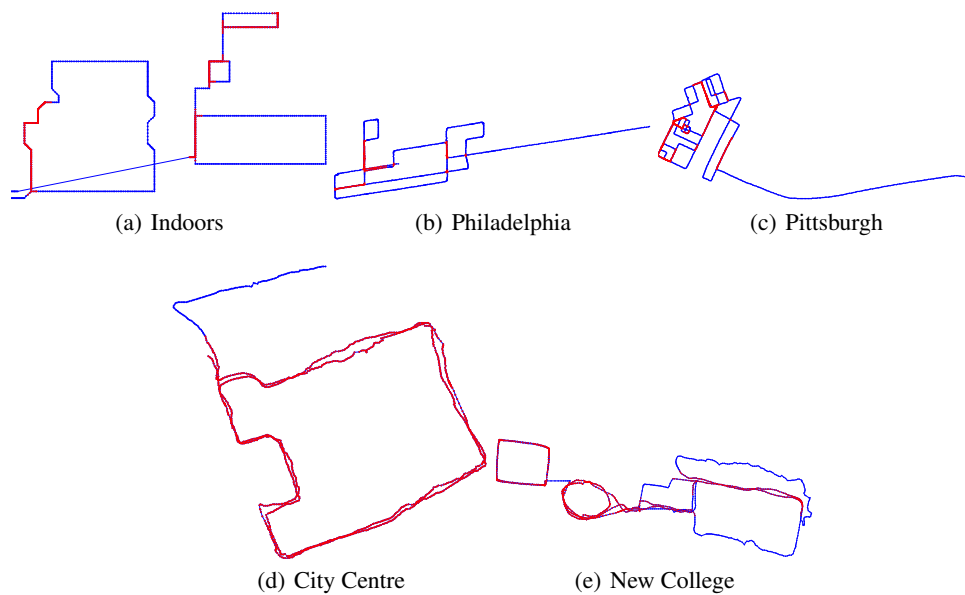

Figure 4: Ground truth maps and loop-closures. Blue dots represent positions of keyframes $K$ with edges $E_T$ drawn in blue. Red dots mark keyframes with loop-closures.

## Footnotes

[1]The Pittsburgh dataset has been provided by Google for research purposes

# References

[1] A. Angeli, D. Filliat, S. Doncieux, and J.-A. Meyer. Fast and incremental method for loop-closure detection using bags of visual words. *Robotics, IEEE Transactions on*, 24(5):1027–1037, Oct. 2008.

[2] Christopher M. Bishop. *Pattern Recognition and Machine Learning (Information Science and Statistics)*. Springer, August 2006.

[3] O. Booij, B. Terwijn, Z. Zivkovic, and B. Krose. Navigation using an appearance based topological map. In *2007 IEEE International Conference on Robotics and Automation*, pages 3927–3932, 2007.

[4] O. Booij, Z. Zivkovic, and B. Krose. Pruning the image set for appearance based robot localization. In *In Proceedings of the Annual Conference of the Advanced School for Computing and Imaging*, 2005.

[5] M. Bosse, P. Newman, J. Leonard, M. Soika, W. Feiten, and S. Teller. An atlas framework for scalable mapping. In *IEEE International Conference on Robotics and Automation, 2003. Proceedings. ICRA'03*, volume 2, 2003.

[6] V. Chvatal. A greedy heuristic for the set-covering problem. *Mathematics of Operations Research*, 4(3):233–235, 1979.

[7] M. Cummins and P. Newman. Accelerated appearance-only SLAM. In *Proc. IEEE International Conference on Robotics and Automation (ICRA'08)*, Pasadena,California, April 2008.

[8] M. Cummins and P. Newman. FAB-MAP: Probabilistic Localization and Mapping in the Space of Appearance. *The International Journal of Robotics Research*, 27(6):647–665, 2008.

[9] F. Fraundorfer, C. Wu, J.-M. Frahm, and M. Pollefeys. Visual word based location recognition in 3d models using distance augmented weighting. In *Fourth International Symposium on 3D Data Processing, Visualization and Transmission*, 2008.

[10] T. Goedemé, M. Nuttin, T. Tuytelaars, and L. Van Gool. Omnidirectional vision based topological navigation. *Int. J. Comput. Vision*, 74(3):219–236, 2007.

[11] D. Koller and N. Friedman. *Probabilistic Graphical Models: Principles and Techniques*. MIT Press, 2009.

[12] D. Lowe. Distinctive image features from scale-invariant keypoints. *International Journal of Computer Vision*, 60:91–110, 2004.

[13] D. Nister and H. Stewenius. Scalable recognition with a vocabulary tree. volume 2, pages 2161–2168, 2006.

[14] A. Ranganathan, E. Menegatti, and F. Dellaert. Bayesian inference in the space of topological maps. *IEEE Transactions on Robotics*, 22(1):92–107, 2006.

[15] D. Scaramuzza, N. Criblez, A. Martinelli, and R. Siegwart. Robust feature extraction and matching for omnidirectional images. Springer Tracts in Advanced Robotics, Field and Service Robotics, 2008.

[16] J.-P. Tardif, Y. Pavlidis, and K. Daniilidis. Monocular visual odometry in urban environments using an omnidirectional camera. pages 2531–2538, Sept. 2008.

[17] N. Tomatis, I. Nourbakhsh, and R. Siegwart. Hybrid simultaneous localization and map building: a natural integration of topological and metric. *Robotics and Autonomous Systems*, 44(1):3–14, 2003.

[18] C. Valgren, T. Duckett, and A. J. Lilienthal. Incremental spectral clustering and its application to topological mapping. In *Proc. IEEE Int. Conf. on Robotics and Automation*, pages 4283–4288, 2007.

[19] C. Valgren, A. J. Lilienthal, and T. Duckett. Incremental topological mapping using omnidirectional vision. In *Proc. IEEE Int. Conf. On Intelligent Robots and Systems*, pages 3441–3447, 2006.

[20] A. Vedaldi and B. Fulkerson. VLFeat: An open and portable library of computer vision algorithms. `http://www.vlfeat.org/`, 2008.

